# Mechanisms of generalization in perceptual learning

**Zili Liu**
Rutgers University, Newark

Daphna Weinshall
Hebrew University, Israel

## Abstract

The learning of many visual perceptual tasks has been shown to be specific to practiced stimuli, while new stimuli require re-learning from scratch. Here we demonstrate generalization using a novel paradigm in motion discrimination where learning has been previously shown to be specific. We trained subjects to discriminate the directions of moving dots, and verified the previous results that learning does not transfer from the trained direction to a new one. However, by tracking the subjects' performance across time in the new direction, we found that their rate of learning doubled. Therefore, learning generalized in a task previously considered too difficult for generalization. We also replicated, in the second experiment, transfer following training with "easy" stimuli.

The specificity of perceptual learning and the dichotomy between learning of "easy" vs. "difficult" tasks were hypothesized to involve different learning processes, operating at different visual cortical areas. Here we show how to interpret these results in terms of signal detection theory. With the assumption of limited computational resources, we obtain the observed phenomena — direct transfer and change of learning rate — for increasing levels of task difficulty. It appears that human generalization concurs with the expected behavior of a generic discrimination system.

## 1 Introduction

Learning in biological systems is of great importance. But while cognitive learning (or "problem solving") is typically abrupt and generalizes to analogous problems, perceptual skills appear to be acquired gradually and specifically: Human subjects cannot generalize a perceptual discrimination skill to solve similar problems with different attributes. For example, in a visual discrimination task (Fig. 1), a subject who is trained to discriminate motion directions between 43° and 47° cannot use

this skill to discriminate 133° from 137°. Generalization has been found only when stimuli of different attributes are interleaved [7, 10], or when the task is easier [6, 1]. For example, a subject who is trained to discriminate 41° from 49° can later readily discriminate 131° from 139° [6]. The specificity of learning has been so far used to support the hypothesis that perceptual learning embodies neuronal modifications in the brain's stimulus-specific cortical areas (e.g., visual area MT) [9, 3, 2, 5, 8, 4].

In contrast to previous results of learning specificity, we show in two experiments in Section 2 that learning in motion discrimination generalizes in all cases where specificity was thought to exist, although the mode of generalization varies. (1) When the task is difficult, it is direction specific in the traditional sense; but learning in a new direction accelerates. (2) When the task is easy, it generalizes to all directions after training in only one direction. While (2) is consistent with the findings reported in [6, 1], (1) demonstrate that generalization is the rule, not an exception limited only to "easy" stimuli.

## 2 Perceptual learning experiments

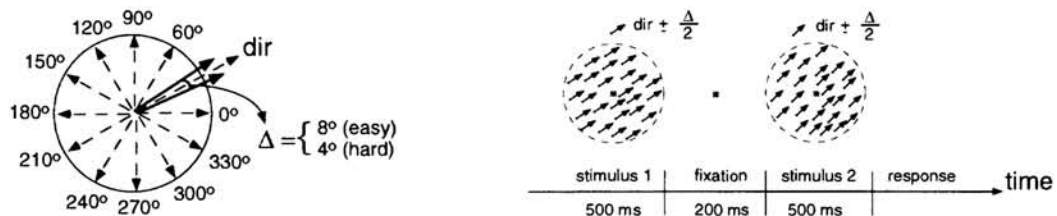

Figure 1: Schematic of one trial. **Left**: the stimulus was a random dot pattern viewed in a circular aperture, spanning 8° of visual angle, moving in a given primary direction (denoted *dir*). The primary direction was chosen from 12 directions, separated by 30°. **Right**: the direction of each of the two stimuli was randomly chosen from two candidate directions ($dir \pm \Delta/2$). The subject judged whether the two stimuli moved in the same or different directions. Feedback was provided.

The motion discrimination task is described in Fig. 1. In each trial, the subject was presented with two consecutive stimuli, each moving in one of two possible directions (randomly chosen from the two directions $dir + \Delta/2$ and $dir - \Delta/2$). The directional difference $|\Delta|$ between the two stimuli was 8° in the easy condition, and 4° in the difficult condition. The experiment was otherwise identical to that in [2] that used $|\Delta| = 3°$, except that our stimuli were displayed on an SGI computer monitor. $|\Delta| = 8°$ was chosen as the easy condition because most subjects found it relatively easy to learn, yet still needed substantial training.

### 2.1 A difficult task

We trained subjects extensively in one primary direction with a difficult motion discrimination task ($\Delta = 4°$), followed by extensive training in a second primary direction. The two primary directions were sufficiently different so direct transfer between them was not expected [2] (Fig. 2). Subjects' initial performance in both directions was comparable, replicating the classical result of stimulus specific learning (no direct transfer). However, all subjects took only half as many training sessions to make the same improvement in the second direction. All subjects had extensive practice with the task prior to this experiment, thus the acceleration cannot be simply explained by familiarity.

Our results show that although perceptual learning did not directly transfer in this difficult task, it did nevertheless generalize to the new direction. The generalization was manifested as 100% increase in the rate of learning in the second direction. It demonstrates that the generalization of learning, as manifested via direct transfer and via increase in learning rate, may be thought of as two extremes of a continuum of possibilities.

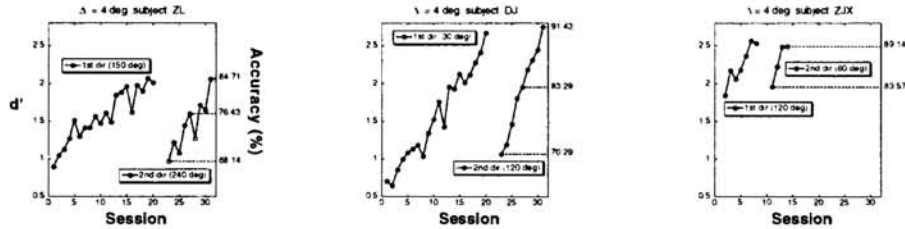

Figure 2: Subjects DJ and ZL needed 20 training sessions in the first direction, and nine in the second; subject ZJX needed seven training sessions in the first, and four in the second. The rate of learning (the amount of improvement per session) in the second direction is significantly greater than in the first ($t(2) = 13.41, p < 0.003$).

## 2.2 An easy task

We first measured the subjects' baseline performance in an easy task — the discrimination of motion directions 8° apart — in 12 primary directions (64 trials each, randomly interleaved). We then trained four subjects in one oblique primary direction (chosen randomly and counter-balanced among subjects) for four sessions, each with 700 trials. Finally, we measured again the subjects' performance in all directions. Every subject improved in all directions (Fig. 3). The performance of seven control subjects was measured without intermediate training; two more control subjects were added who were "trained" with similar motion stimuli but were asked to discriminate a brightness change instead. The control subjects improved as well, but significantly less ($\Delta d' = 0.09$ vs. 0.78, Fig. 3).

Our results clearly show that training with an easy task in one direction leads to immediate improvement in other directions. Hence the learned skill generalized across motion directions.

## 3 A computational model

We will now adopt a general framework for the analysis of perceptual learning results, using the language of signal detection theory. Our model accounts for the results in this paper by employing the constraint of limited computational resources. The model's assumptions are as follows.

1. In each trial, each of the two stimuli is represented by a population of measurements that encode all aspects of the stimulus, in particular, the output of localized direction detectors. The measurements are encoded as a vector. The decision as to whether the two stimuli are the same or not is determined by the difference of the two vectors.

2. Each component of the input measurements is characterized by its sensitivity for the discrimination task, e.g., how well the two motion directions can be discriminated apart based on this component. The entire population itself is generally divided into two sets: *informative* — measurements with significant sensitivity, and

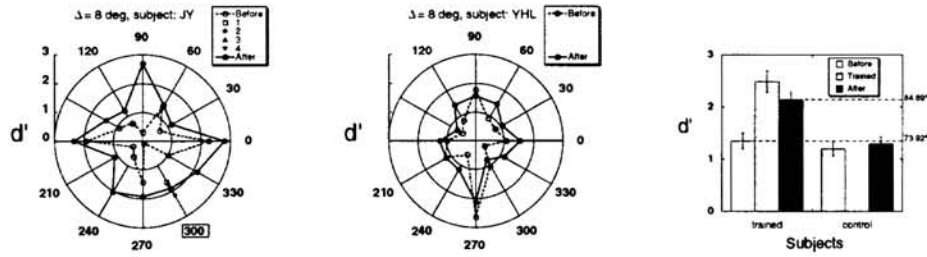

Figure 3: **Left**: Discrimination sensitivity $d'$ of subject JY who was trained in the primary direction 300°. **Middle**: $d'$ of control subject YHL who had no training in between the two measurements. **Right**: Average $d'$ (and standard error) for all subjects before and after training. **Trained**: results for the four trained subjects. Note the substantial improvement between the two measurements. For these subjects, the $d'$ measured after training is shown separately for the trained direction (middle column) and the remaining directions (right column). **Control**: results for the nine control subjects. The control subjects improved their performance significantly less than the trained subjects ($\Delta d' = 0.09$ vs. $0.78$ ; $F(1, 11) = 14.79, p < 0.003$).

*uninformative* — measurements with null sensitivity. In addition, *informative* measurements may vary greatly in their individual sensitivity. When many have high sensitivity, the task is easy. When most have low sensitivity, the task is difficult.

We assume that sensitivity changes from one primary direction to the next, but the population of *informative* measurements remains constant. For example, in our psychophysical task localized directional signals are likely to be in the *informative* set for any motion direction, though their individual sensitivity will vary based on specific motion directions. On the other hand, local speed signals are never informative and therefore always belong to the *uninformative* set.

3. Due to limited computational capacity, the system can, at a time, only process a small number of components of the input vector. The decision in a single trial is therefore made based on the magnitude of this sub-vector, which may vary from trial to trial.

In each trial the system rates the processed components of the sub-vector according to their sensitivity for the discrimination task. After a sufficient number of trials (enough to estimate all the component sensitivities of the sub-vector), the system identifies the least sensitive component and replaces it in the next trial with a new random component from the input vector. In effect, the system is searching from the input vector a sub-vector that gives rise to the maximal discrimination sensitivity. Therefore the performance of the system is gradually improving, causing learning from session to session in the training direction.

4. After learning in one training direction, the system identifies the sets of *informative* and *uninformative* measurements and include in the *informative* set any measurement with significant (though possibly low) sensitivity. In the next training direction, only the set of *informative* measurements is searched. The search becomes more efficient, and hence the acceleration of the learning rate. This accounts for the learning between training directions.

We further assume that each stimulus generates a signal that is a vector of $N$ measurements: $\{I_i\}_{i=1}^{N}$. We also assume that the signal for the discrimination task is the difference between two stimulus measurements: $\mathbf{x} = \{x_i\}_{i=1}^{N}$, $x_i = \Delta I_i$. The

same/different discrimination task is to decide whether $\mathbf{x}$ is generated by noise — the null vector $\emptyset$, or by some distinct signal — the vector $\mathcal{S}$.

At time $t$ a measurement vector $\mathbf{x}^t$ is obtained, which we denote $\mathbf{x}^{st}$ if it is the signal $\mathcal{S}$, and $\mathbf{x}^{nt}$ otherwise. Assume that each measurement in $\mathbf{x}^t$ is a normal random variable: $\mathbf{x}^{nt} = \{x_i^{nt}\}_{i=1}^N, x_i^{nt} \sim N(0, \sigma_i), \mathbf{x}^{st} = \{x_i^{st}\}_{i=1}^N, x_i^{st} \sim N(\mu_i, \sigma_i)$. We measure the sensitivity $d'$ of each component. Since both the signal and noise are assumed to be normal random variables, the sensitivity of the $i$-th measurement in the discrimination task is $d'_i = |\mu_i|/\sigma_i$. Assuming further that the measurements are independent of each other and of time, then the combined sensitivity of $M$ measurements is $d' = \sqrt{\sum_{i=1}^M (\mu_i/\sigma_i)^2}$.

## 3.1 Limited resources: an assumption

We assume that the system can simultaneously process at most $M \ll N$ of the original $N$ measurements. Since the sensitivity $d'_i$ of the different measurements varies, the discrimination depends on the combined sensitivity of the particular set of $M$ measurements that are being used. Learning in the first training direction, therefore, leads to the selection of a "good" subset of the measurements, obtained by searching in the measurement space.

After searching for the best $M$ measurements for the current training direction, the system divides the measurements into two sets: those with non-negligible sensitivity, and those with practically null sensitivity. This rating is kept for the next training direction, when only the first set is searched.

One prediction of this model is that learning rate should *not* increase with exposure only. In other words, it is necessary for subjects to be exposed to the stimulus **and** do the same discrimination task for effective inter-directional learning to take place. For example, assume that the system is given $N$ measurements: $N/2$ motion direction signals and $N/2$ speed signals. It learns during the first training direction that the $N/2$ speed signals have null sensitivity for the direction discrimination task, whereas the directional signals have varying (but significant) sensitivity. In the second training direction, the system is given the $N$ measurements whose sensitivity profile is different from that in the first training direction, but still with the property that only the directional signals have any significant sensitivity (Fig. 4b). Based on learning in the first training direction, the system only searches the measurements whose sensitivity in the first training direction was significant, namely, the $N/2$ directional signals. It ignores the speed signals. Now the asymptotic performance in the second direction remains unchanged because the most sensitive measurements are within the searched population — they are directional signals. The learning rate, however, doubles since the system searches a space half as large.

## 3.2 Simulation results

To account for the different modes of learning, we make the following assumptions. When the task is easy, many components have high sensitivity $d'$. When the task is difficult, only a small number of measurements have high $d'$. Therefore, when the task is easy, a subset of $M$ measurements that give rise to the best performance is found relatively fast. In the extreme, when the task is very easy (e.g., all the measurements have very high sensitivity), the rate of learning is almost instantaneous and the observed outcome appears to be transfer. On the other hand, when the task is difficult, it takes a long time to find the $M$ measurements that give rise to the best performance, and learning is slow.

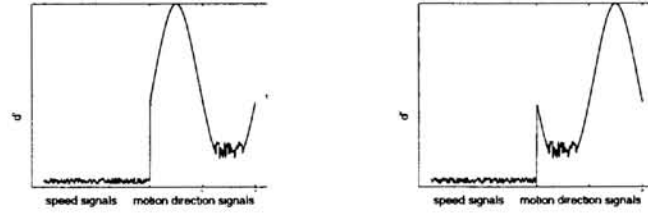

Figure 4: Hypothetical sensitivity profile for a population of measurements of speed and motion direction. **Left**: First training direction — only the motion direction measurements have significant sensitivity ($d'$ above 0.1), with measurements around 45° having the highest $d'$. **Right**: Second direction — only the motion direction measurements have significant sensitivity, with measurements around 135° having the highest $d'$.

The detailed operations of the model are as follows. In the first training direction, the system starts with a random set of $M$ measurements. In each trial and using feedback, the mean and standard deviation of each measurement is computed: $\mu_i^{st}$, $\sigma_i^{st}$ for the signal and $\mu_i^{nt}$, $\sigma_i^{nt}$ for the noise. In the next trial, given $M$ measurements $\{x_i^{t+1}\}_{i=1}^M$, the system evaluates $\delta = \sum_{i=1}^M \left(\frac{x_i^{t+1}-\mu_i^{st}}{\sigma_i^{st}}\right)^2 - \left(\frac{x_i^{t+1}-\mu_i^{nt}}{\sigma_i^{nt}}\right)^2$ , and classifies $\mathbf{x}$ as the signal if $\delta < 0$, and noise otherwise.

At time $T$, the worst measurement is identified as *argval* of $\min_i d_i'$, $d_i' = 2|\mu_i^{sT} - \mu_i^{nT}|/(\sigma_i^{st} + \sigma_i^{nt})$. It is then replaced randomly from one of the remaining $N - M$ measurements. The learning and decision making then proceed as above for another $T$ iterations. This is repeated until the set of chosen measurements stabilizes. At the end, the decision is made based on the set of $M$ measurements that have the highest sensitivities.

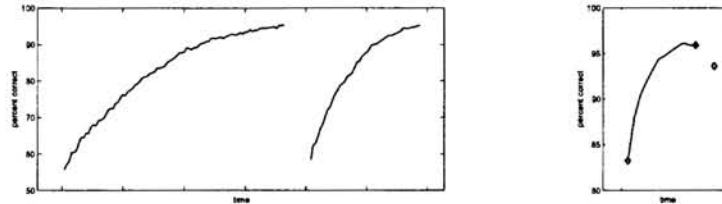

Figure 5: Simulated performance (percent correct) as function of time. **Left**: Difficult condition — the number of measurements with high $d_i'$ is small (4 out of 150); there is no transfer from the first to the second training direction, but the learning rate is increased two-fold. This graph is qualitatively similar to the results shown in the top row of Fig. 2. **Right**: Easy condition — the number of measurements with high $d_i'$ is large (72 out of 150); there is almost complete transfer from the first to the second training direction.

At the very beginning of training in the second direction, based on the measured $d_i'$ in the first direction, the measurement population is labeled as *informative* — those with $d_i'$ larger than the median value, and *uninformative* — the remaining measurements. The learning and decision making proceeds as above, while only *informative* measurements are considered during the search.

In the simulation we used $N = 150$ measurements, with $M = 4$. Half of the $N$ measurements (the *informative* measurements) had significant $d_i'$. In the second training direction, the sensitivities of the measurements were randomly changed, but only the *informative* measurements had significant $d_i'$. By varying the number of measurements with high $d_i'$ in the population of *informative* measurements, we get the different modes of generalization(Fig. 5).

## 4 Discussions

In contrast to previous results on the specificity of learning, we broadened the search for generalization beyond traditional transfer. We found that generalization is the rule, rather than an exception. Perceptual learning of motion discrimination generalizes in various forms: as acceleration of learning rate (Exp. 1), as immediate improvement in performance (Exp. 2). Thus we show that perceptual learning is more similar to cognitive learning than previously thought, with both stimulus specificity and generalization as important ingredients.

In our scheme, the assumption of the computational resource forced the discrimination system to search in the measurement space. The generalization phenomena — transfer and increased learning rate — occur due to improvement in *search sensitivity* from one training direction to the next, as the size of the search space decreases with learning. Our scheme also predicts that learning rate should *only* improve if the subject both sees the stimulus **and** does the relevant discrimination task, in agreement with the results in Exp. 1. Importantly, our scheme does not predict transfer *per se*, but instead a dramatic increase in learning rate that is *equivalent* to transfer.

Our model is qualitative and does not make any concrete quantitative predictions. We would like to emphasize that this is not a handicap of the model. Our goal is to show, qualitatively, that the various generalization phenomena should not surprise us, as they should naturally occur in a generic discrimination system with limited computational resources. Thus we argue that it may be too early to use existing perceptual learning results for the identification of the cortical location of perceptual learning, and the levels at which modifications are taking place.

## References

[1] Ahissar M and Hochstein S. Task difficulty and the specificity of perceptual learning. *Nature*, 387:401–406, 1997.

[2] Ball K and Sekuler R. A specific and enduring improvement in visual motion discrimination. *Science*, 218:697–698, 1982.

[3] Fiorentini A and Berardi N. Perceptual learning specific for orientation and spatial frequency. *Nature*, 287:43–44, 1980.

[4] Gilbert C D. Early perceptual learning. *PNAS*, 91:1195–1197, 1994.

[5] Karni A and Sagi D. Where practice makes perfect in texture discrimination: Evidence for primary visual cortex plasticity. *PNAS*, 88:4966–4970, 1991.

[6] Liu Z. Learning a visual skill that generalizes. Tech. Report, NECI, 1995.

[7] Liu Z and Vaina L M. Stimulus specific learning: a consequence of stimulus-specific experiments? *Perception*, 24(supplement):21, 1995.

[8] Poggio T, Fahle M, and Edelman S. Fast perceptual learning in visual hyper-acuity. *Science*, 256:1018–1021, May 1992.

[9] Ramachandran V S. Learning-like phenomena in stereopsis. *Nature*, 262:382–384, 1976.

[10] Rubin N, Nakayama K, and Shapley R. Abrupt learning and retinal size specificity in illusory-contour perception. *Current Biology*, 7:461–467, 1997.